# Binary Coding in Auditory Cortex

**Michael R. DeWeese and Anthony M. Zador**
Cold Spring Harbor Laboratory, Cold Spring Harbor, NY 11724
*deweese@cshl.edu, zador@cshl.edu*

## Abstract

Cortical neurons have been reported to use both rate and temporal codes. Here we describe a novel mode in which each neuron generates exactly 0 or 1 action potentials, but not more, in response to a stimulus. We used cell-attached recording, which ensured single-unit isolation, to record responses in rat auditory cortex to brief tone pips. Surprisingly, the majority of neurons exhibited binary behavior with few multi-spike responses; several dramatic examples consisted of exactly one spike on 100% of trials, with no trial-to-trial variability in spike count. Many neurons were tuned to stimulus frequency. Since individual trials yielded at most one spike for most neurons, the information about stimulus frequency was encoded in the population, and would not have been accessible to later stages of processing that only had access to the activity of a single unit. These binary units allow a more efficient population code than is possible with conventional rate coding units, and are consistent with a model of cortical processing in which synchronous packets of spikes propagate stably from one neuronal population to the next.

## 1  Binary coding in auditory cortex

We recorded responses of neurons in the auditory cortex of anesthetized rats to pure-tone pips of different frequencies [1, 2]. Each pip was presented repeatedly, allowing us to assess the variability of the neural response to multiple presentations of each stimulus. We first recorded multi-unit activity with conventional tungsten electrodes (*Fig. 1a*). The number of spikes in response to each pip fluctuated markedly from one trial to the next (*Fig. 1e*), as though governed by a random mechanism such as that generating the ticks of a Geiger counter. Highly variable responses such as these, which are at least as variable as a Poisson process, are the norm in the cortex [3-7], and have contributed to the widely held view that cortical spike trains are so noisy that only the average firing rate can be used to encode stimuli.

Because we were recording the activity of an unknown number of neurons, we could not be sure whether the strong trial-to-trial fluctuations reflected the underlying variability of the single units. We therefore used an alternative technique, cell-

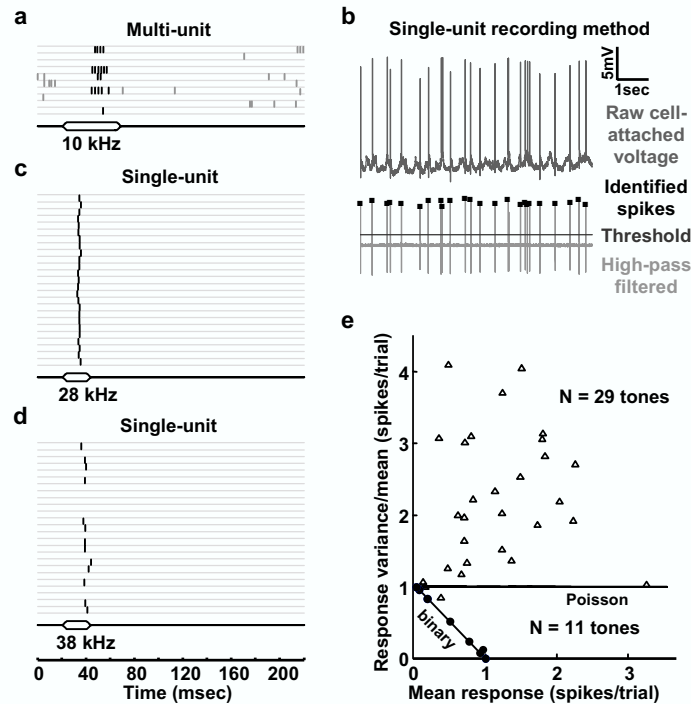

**Figure 1:** Multi-unit spiking activity was highly variable, but single units obeyed binomial statistics. **a** Multi-unit spike rasters from a conventional tungsten electrode recording showed high trial-to-trial variability in response to ten repetitions of the same 50 msec pure tone stimulus (*bottom*). Darker hash marks indicate spike times within the response period, which were used in the variability analysis. **b** Spikes recorded in cell-attached mode were easily identified from the raw voltage trace (*top*) by applying a high-pass filter (*bottom*) and thresholding (*dark gray line*). Spike times (*black squares*) were assigned to the peaks of suprathreshold segments. **c** Spike rasters from a cell-attached recording of single-unit responses to 25 repetitions of the same tone consisted of exactly one well-timed spike per trial (latency standard deviation = 1.0 msec), unlike the multi-unit responses (*Fig. 1a*). Under the Poisson assumption, this would have been highly unlikely ($P \sim 10^{-11}$). **d** The same neuron as in Fig. 1c responds with lower probability to repeated presentations of a different tone, but there are still no multi-spike responses. **e** We quantified response variability for each tone by dividing the variance in spike count by the mean spike count across all trials for that tone. Response variability for multi-unit tungsten recording (*open triangles*) was high for each of the 29 tones (out of 32) that elicited at least one spike on one trial. All but one point lie above one (*horizontal gray line*), which is the value produced by a Poisson process with any constant or time varying event rate. Single unit responses recorded in cell-attached mode were far less variable (*filled circles*). Ninety one percent (10/11) of the tones that elicited at least one spike from this neuron produced no multi-spike responses in 25 trials; the corresponding points fall on the diagonal line between (0,1) and (1,0), which provides a strict lower bound on the variability for any response set with a mean between 0 and 1. No point lies above one.

attached recording with a patch pipette [8, 9], in order to ensure single unit isolation (*Fig. 1b*). This recording mode minimizes both of the main sources of error in spike detection: failure to detect a spike in the unit under observation (false negatives), and contamination by spikes from nearby neurons (false positives). It also differs from conventional extracellular recording methods in its selection bias: With cell-

attached recording neurons are selected solely on the basis of the experimenter's ability to form a seal, rather than on the basis of neuronal activity and responsiveness to stimuli as in conventional methods.

Surprisingly, single unit responses were far more orderly than suggested by the multi-unit recordings; responses typically consisted of either 0 or 1 spikes per trial, and not more (*Fig. 1c-e*). In the most dramatic examples, each presentation of the same tone pip elicited exactly one spike (*Fig. 1c*). In most cases, however, some presentations failed to elicit a spike (*Fig. 1d*). Although low-variability responses have recently been observed in the cortex [10, 11] and elsewhere [12, 13], the binary behavior described here has not previously been reported for cortical neurons.

The majority of the neurons (59%) in our study for which statistical significance could be assessed (at the *p<0.001 significance level; see Fig. 2, caption*) showed noisy binary behavior—"binary" because neurons produced either 0 or 1 spikes, and "noisy" because some stimuli elicited both single spikes and failures. In a substantial fraction of neurons, however, the responses showed more variability. We found no correlation between neuronal variability and cortical layer (inferred from the depth of the recording electrode), cortical area (inside vs. outside of area A1) or depth of anesthesia. Moreover, the binary mode of spiking was not due to the brevity (25 msec) of the stimuli; responses that were binary for short tones were comparably binary when longer (100 msec) tones were used (*Fig. 2b*).

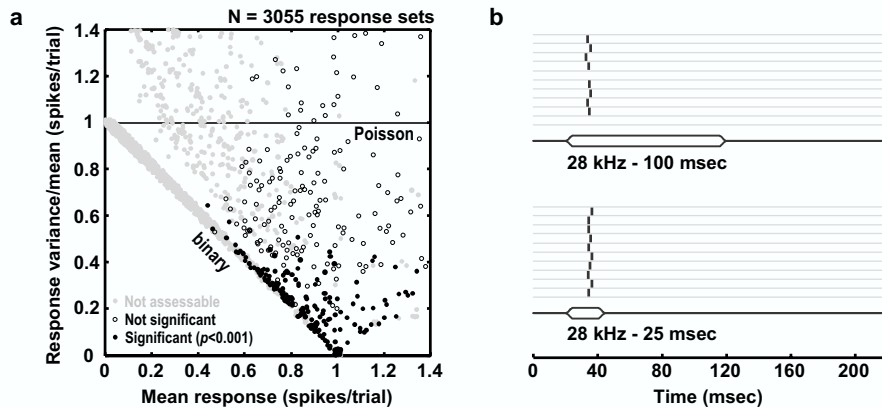

**Figure 2:** Half of the neuronal population exhibited binary firing behavior. **a** Of the 3055 sets of responses to 25 msec tones, 2588 (*gray points*) *could not be assessed* for significance at the *p<0.001* level, 225 (*open circles*) were *not significantly binary*, and 242 were *significantly binary* (*black points;* see *Identification methods for group statistics* below). All points were jittered slightly so that overlying points could be seen in the figure. 2165 response sets contained no multi-spike responses; the corresponding points fell on the line from [0,1] to [1,0]. **b** The binary nature of single unit responses was insensitive to tone duration, even for frequencies that elicited the largest responses. Twenty additional spike rasters from the same neuron (and tone frequency) as in Fig. 1c contain no multi-spike responses whether in response to 100 msec tones (*above*) or 25 msec tones (*below*). Across the population, binary responses were as prevalent for 100 msec tones as for 25 msec tones (see *Identification methods for group statistics*).

In many neurons, binary responses showed high temporal precision, with latencies sometimes exhibiting standard deviations as low as 1 msec (Fig. 3; see also Fig. 1c), comparable to previous observations in the auditory cortex [14], and only slightly

more precise than in monkey visual area MT [5]. High temporal precision was positively correlated with high response probability (Fig. 3).

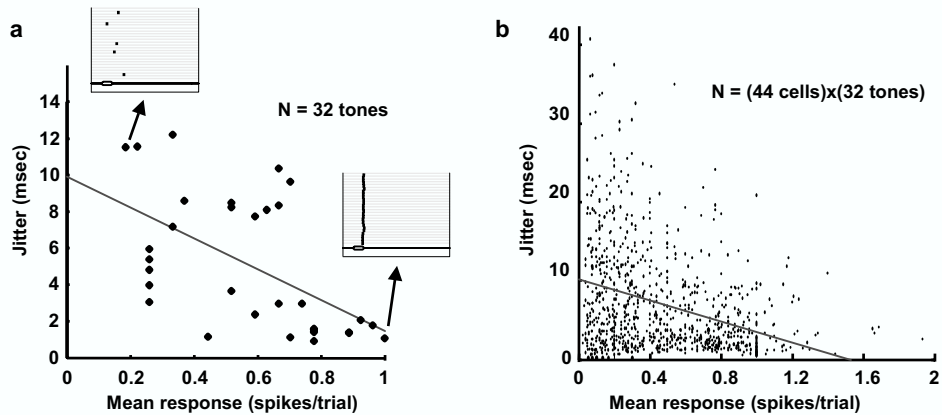

**Figure 3:** Trial-to-trial variability in latency of response to repeated presentations of the same tone decreased with increasing response probability. **a** Scatter plot of standard deviation of latency vs. mean response for 25 presentations each of 32 tones for a different neuron as in Figs. 1 and 2 (*gray line* is best linear fit). Rasters from 25 repeated presentations of a low response tone (*upper left inset*, which corresponds to *left-most data point*) display much more variable latencies than rasters from a high response tone (*lower right inset*; corresponds to *right-most data point*). **b** The negative correlation between latency variability and response size was present on average across the population of 44 neurons described in *Identification methods for group statistics* (linear fit, *gray*).

The low trial-to-trial variability ruled out the possibility that the firing statistics could be accounted for by a simple rate-modulated Poisson process (*Fig. 4a1,a2*). In other systems, low variability has sometimes been modeled as a Poisson process followed by a post-spike refractory period [10, 12]. In our system, however, the range in latencies of evoked binary responses was often much greater than the refractory period, which could not have been longer than the 2 msec inter-spike intervals observed during epochs of spontaneous spiking, indicating that binary spiking did not result from any intrinsic property of the spike generating mechanism (*Fig. 4a3*). Moreover, a single stimulus-evoked spike could suppress subsequent spikes for as long as hundreds of milliseconds (*e.g. Figs. 1d,4d*), supporting the idea that binary spiking arises through a circuit-level, rather than a single-neuron, mechanism. Indeed, the fact that this suppression is observed even in the cortex of awake animals [15] suggests that binary spiking is not a special property of the anesthetized state.

It seems surprising that binary spiking in the cortex has not previously been remarked upon. In the auditory cortex the explanation may be in part technical: Because firing rates in the auditory cortex tend to be low, multi-unit recording is often used to maximize the total amount of data collected. Moreover, our use of cell-attached recording minimizes the usual bias toward responsive or active neurons.

Such explanations are not, however, likely to account for the failure to observe binary spiking in the visual cortex, where spike count statistics have been scrutinized more closely [3-7]. One possibility is that this reflects a fundamental difference between the auditory and visual systems. An alternative interpretation—

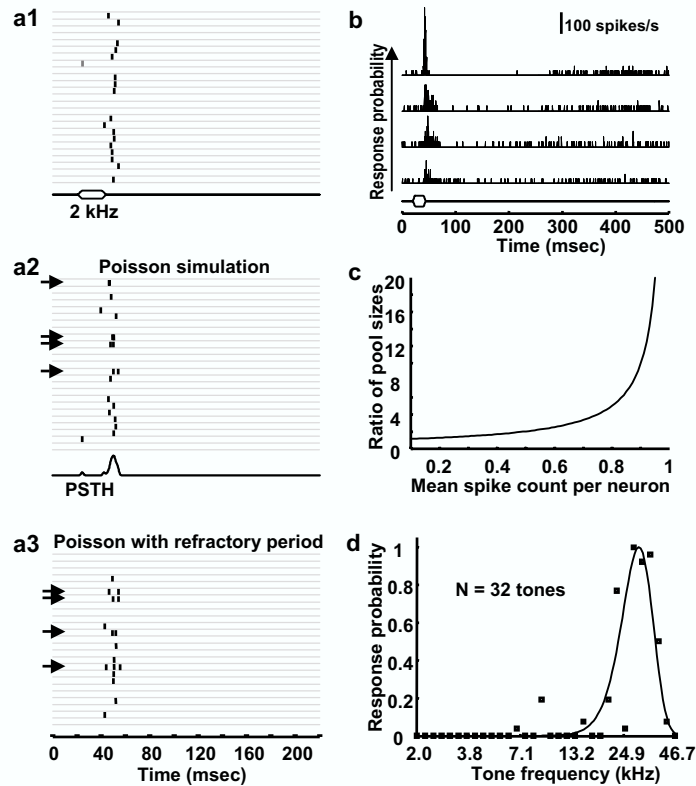

**Figure 4: a** The lack of multi-spike responses elicited by the neuron shown in Fig. 3a were not due to an absolute refractory period since the range of latencies for many tones, like that shown here, was much greater than any reasonable estimate for the neuron's refractory period. **(a1)** Experimentally recorded responses. **(a2)** Using the smoothed post stimulus time histogram (PSTH; *bottom*) from the set of responses in Fig. 4a, we generated rasters under the assumption of Poisson firing. In this representative example, four double-spike responses (*arrows at left*) were produced in 25 trials. **(a3)** We then generated rasters assuming that the neuron fired according to a Poisson process subject to a hard refractory period of 2 msec. Even with a refractory period, this representative example includes one triple- and three double-spike responses. The minimum interspike-interval during spontaneous firing events was less than two msec for five of our neurons, so 2 msec is a conservative upper bound for the refractory period. **b.** Spontaneous activity is reduced following high-probability responses. The PSTH (top; 0.25 msec bins) of the combined responses from the 25% (8/32) of tones that elicited the largest responses from the same neuron as in Figs. 3a and 4a illustrates a preclusion of spontaneous and evoked activity for over 200 msec following stimulation. The PSTHs from progressively less responsive groups of tones show progressively less preclusion following stimulation. **c** Fewer noisy binary neurons need to be pooled to achieve the same "signal-to-noise ratio" (SNR; see ref. [24]) as a collection of Poisson neurons. The ratio of the number of Poisson to binary neurons required to achieve the same SNR is plotted against the mean number of spikes elicited per neuron following stimulation; here we have defined the SNR to be the ratio of the mean spike count to the standard deviation of the spike count. **d** Spike probability tuning curve for the same neuron as in Figs. 1c-e and 2b fit to a Gaussian in tone frequency.

and one that we favor—is that the difference rests not in the sensory modality, but instead in the difference between the stimuli used. In this view, the binary responses may not be limited to the auditory cortex; neurons in visual and other sensory cortices might exhibit similar responses to the appropriate stimuli. For example, the

tone pips we used might be the auditory analog of a brief flash of light, rather than the oriented moving edges or gratings usually used to probe the primary visual cortex. Conversely, auditory stimuli analogous to edges or gratings [16, 17] may be more likely to elicit conventional, rate-modulated Poisson responses in the auditory cortex. Indeed, there may be a continuum between binary and Poisson modes. Thus, even in conventional rate-modulated responses, the first spike is often privileged in that it carries most of the information in the spike train [5, 14, 18]. The first spike may be particularly important as a means of rapidly signaling stimulus transients.

Binary responses suggest a mode that complements conventional rate coding. In the simplest rate-coding model, a stimulus parameter (such as the frequency of a tone) governs only the rate at which a neuron generates spikes, but not the detailed positions of the spikes; the actual spike train itself is an instantiation of a random process (such as a Poisson process). By contrast, in the binomial model, the stimulus parameter (frequency) is encoded as the *probability* of firing (*Fig. 4d*).

Binary coding has implications for cortical computation. In the rate coding model, stimulus encoding is "ergodic": a stimulus parameter can be read out either by observing the activity of one neuron for a long time, or a population for a short time. By contrast, in the binary model the stimulus value can be decoded only by observing a neuronal population, so that there is no benefit to integrating over long time periods (cf. ref. [19]). One advantage of binary encoding is that it allows the population to signal quickly; the most compact message a neuron can send is one spike [20]. Binary coding is also more efficient in the context of population coding, as quantified by the signal-to-noise ratio (*Fig. 4c*).

The precise organization of both spike number and time we have observed suggests that cortical activity consists, at least under some conditions, of packets of spikes synchronized across populations of neurons. Theoretical work [21-23] has shown how such packets can propagate stably from one population to the next, but only if neurons within each population fire at most one spike per packet; otherwise, the number of spikes per packet—and hence the width of each packet—grows at each propagation step. Interestingly, one prediction of stable propagation models is that spike probability should be related to timing precision, a prediction born out by our observations (*Fig. 3*). The role of these packets in computation remains an open question.

## 2   Identification methods for group statistics

We recorded responses to 32 different 25 msec tones from each of 175 neurons from the auditory cortices of 16 Sprague-Dawley rats; each tone was repeated between 5 and 75 times (mean = 19). Thus our ensemble consisted of 32x175=5600 response sets, with between 5 and 75 samples in each set. Of these, 3055 response sets contained at least one spike on at least on trial. For each response set, we tested the hypothesis that the observed variability was significantly lower than expected from the null hypothesis of a Poisson process. The ability to assess significance depended on two parameters: the sample size (5-75) and the firing probability. Intuitively, the dependence on firing probability arises because at low firing rates most responses produce only trials with 0 or 1 spikes under both the Poisson and binary models; only at high firing rates do the two models make different predictions, since in that case the Poisson model includes many trials with 2 or even 3 spikes while the binary model generates only solitary spikes (*see Fig. 4a1,a2*). Using a stringent significance criterion of *p<0.001,* 467 response sets had a sufficient number of repeats to assess significance, given the observed firing probability. Of these, half (242/467=52%) were significantly less variable than expected by chance, five hundred-fold higher than the 467/1000=0.467 response sets expected, based on the

0.001 significance criterion, to yield a binary response set. Seventy-two neurons had at least one response set for which significance could be assessed, and of these, 49 neurons (49/72=68%) had at least one significantly sub-Poisson response set. Of this population of 49 neurons, five achieved low variability through repeatable bursty behavior (e.g., every spike count was either 0 or 3, but not 1 or 2) and were excluded from further analysis. The remaining 44 neurons formed the basis for the group statistics analyses shown in Figs. 2a and 3b. Nine of these neurons were subjected to an additional protocol consisting of at least 10 presentations each of 100 msec tones and 25 msec tones of all 32 frequencies. Of the 100 msec stimulation response sets, 44 were found to be significantly sub-Poisson at the $p<0.05$ level, in good agreement with the 43 found to be significant among the responses to 25 msec tones.

## 3   Bibliography

1.      Kilgard, M.P. and M.M. Merzenich, Cortical map reorganization enabled by nucleus basalis activity. *Science*, 1998. **279**(5357): p. 1714-8.

2.      Sally, S.L. and J.B. Kelly, Organization of auditory cortex in the albino rat: sound frequency. *J Neurophysiol*, 1988. **59**(5): p. 1627-38.

3.      Softky, W.R. and C. Koch, The highly irregular firing of cortical cells is inconsistent with temporal integration of random EPSPs. *J Neurosci*, 1993. **13**(1): p. 334-50.

4.      Stevens, C.F. and A.M. Zador, Input synchrony and the irregular firing of cortical neurons. *Nat Neurosci*, 1998. **1**(3): p. 210-7.

5.      Buracas, G.T., A.M. Zador, M.R. DeWeese, and T.D. Albright, Efficient discrimination of temporal patterns by motion-sensitive neurons in primate visual cortex. *Neuron*, 1998. **20**(5): p. 959-69.

6.      Shadlen, M.N. and W.T. Newsome, The variable discharge of cortical neurons: implications for connectivity, computation, and information coding. *J Neurosci*, 1998. **18**(10): p. 3870-96.

7.      Tolhurst, D.J., J.A. Movshon, and A.F. Dean, The statistical reliability of signals in single neurons in cat and monkey visual cortex. *Vision Res*, 1983. **23**(8): p. 775-85.

8.      Otmakhov, N., A.M. Shirke, and R. Malinow, Measuring the impact of probabilistic transmission on neuronal output. *Neuron*, 1993. **10**(6): p. 1101-11.

9.      Friedrich, R.W. and G. Laurent, Dynamic optimization of odor representations by slow temporal patterning of mitral cell activity. *Science*, 2001. **291**(5505): p. 889-94.

10.     Kara, P., P. Reinagel, and R.C. Reid, Low response variability in simultaneously recorded retinal, thalamic, and cortical neurons. *Neuron*, 2000. **27**(3): p. 635-46.

11.     Gur, M., A. Beylin, and D.M. Snodderly, Response variability of neurons in primary visual cortex (V1) of alert monkeys. *J Neurosci*, 1997. **17**(8): p. 2914-20.

12.     Berry, M.J., D.K. Warland, and M. Meister, The structure and precision of retinal spike trains. *Proc Natl Acad Sci U S A*, 1997. **94**(10): p. 5411-6.

13.     de Ruyter van Steveninck, R.R., G.D. Lewen, S.P. Strong, R. Koberle, and W. Bialek, Reproducibility and variability in neural spike trains. *Science*, 1997. **275**(5307): p. 1805-8.

14.     Heil, P., Auditory cortical onset responses revisited. I. First-spike timing. *J Neurophysiol*, 1997. **77**(5): p. 2616-41.

15.     Lu, T., L. Liang, and X. Wang, Temporal and rate representations of time-varying signals in the auditory cortex of awake primates. *Nat Neurosci*, 2001. **4**(11): p. 1131-8.

16.     Kowalski, N., D.A. Depireux, and S.A. Shamma, Analysis of dynamic spectra in ferret primary auditory cortex. I. Characteristics of single-unit responses to moving ripple spectra. *J Neurophysiol*, 1996. **76**(5): p. 3503-23.

17.     deCharms, R.C., D.T. Blake, and M.M. Merzenich, Optimizing sound features for cortical neurons. *Science*, 1998. **280**(5368): p. 1439-43.

18.     Panzeri, S., R.S. Petersen, S.R. Schultz, M. Lebedev, and M.E. Diamond, The role of spike timing in the coding of stimulus location in rat somatosensory cortex. *Neuron*, 2001. **29**(3): p. 769-77.

19.     Britten, K.H., M.N. Shadlen, W.T. Newsome, and J.A. Movshon, The analysis of visual motion: a comparison of neuronal and psychophysical performance. *J Neurosci*, 1992. **12**(12): p. 4745-65.

20.     Delorme, A. and S.J. Thorpe, Face identification using one spike per neuron: resistance to image degradations. *Neural Netw*, 2001. **14**(6-7): p. 795-803.

21.     Diesmann, M., M.O. Gewaltig, and A. Aertsen, Stable propagation of synchronous spiking in cortical neural networks. *Nature*, 1999. **402**(6761): p. 529-33.

22.     Marsalek, P., C. Koch, and J. Maunsell, On the relationship between synaptic input and spike output jitter in individual neurons. *Proc Natl Acad Sci U S A*, 1997. **94**(2): p. 735-40.

23.     Kistler, W.M. and W. Gerstner, Stable propagation of activity pulses in populations of spiking neurons. *Neural Comp.*, 2002. **14**: p. 987-997.

24.     Zohary, E., M.N. Shadlen, and W.T. Newsome, Correlated neuronal discharge rate and its implications for psychophysical performance. *Nature*, 1994. **370**(6485): p. 140-3.

25.     Abbott, L.F. and P. Dayan, The effect of correlated variability on the accuracy of a population code. *Neural Comput*, 1999. **11**(1): p. 91-101.
